# Learning to Control an Octopus Arm with Gaussian Process Temporal Difference Methods

**Yaakov Engel**[*]
AICML, Dept. of Computing Science
University of Alberta
Edmonton, Canada
`yaki@cs.ualberta.ca`

**Peter Szabo and Dmitry Volkinshtein**
Dept. of Electrical Engineering
Technion Institute of Technology
Haifa, Israel
`peter.z.szabo@gmail.com`
`dmitryvolk@gmail.com`

## Abstract

The Octopus arm is a highly versatile and complex limb. How the Octopus controls such a hyper-redundant arm (not to mention eight of them!) is as yet unknown. Robotic arms based on the same mechanical principles may render present day robotic arms obsolete. In this paper, we tackle this control problem using an online reinforcement learning algorithm, based on a Bayesian approach to policy evaluation known as Gaussian process temporal difference (GPTD) learning. Our substitute for the real arm is a computer simulation of a 2-dimensional model of an Octopus arm. Even with the simplifications inherent to this model, the state space we face is a high-dimensional one. We apply a GPTD-based algorithm to this domain, and demonstrate its operation on several learning tasks of varying degrees of difficulty.

## 1  Introduction

The Octopus arm is one of the most sophisticated and fascinating appendages found in nature. It is an exceptionally flexible organ, with a remarkable repertoire of motion. In contrast to skeleton-based vertebrate and present-day robotic limbs, the Octopus arm lacks a rigid skeleton and has virtually infinitely many degrees of freedom. As a result, this arm is highly hyper-redundant – it is capable of stretching, contracting, folding over itself several times, rotating along its axis at any point, and following the contours of almost any object. These properties allow the Octopus to exhibit feats requiring agility, precision and force. For instance, it is well documented that Octopuses are able to pry open a clam or remove the plug off a glass jar, to gain access to its contents [1].

The basic mechanism underlying the flexibility of the Octopus arm (as well as of other organs, such as the elephant trunk and vertebrate tongues) is the muscular hydrostat [2]. Muscular hydrostats are organs capable of exerting force and producing motion with the sole use of muscles. The muscles serve in the dual roles of generating the forces and maintaining the structural rigidity of the appendage. This is possible due to a constant volume constraint, which arises from the fact that muscle tissue is incompressible. Proper

---

[*]To whom correspondence should be addressed. Web site: `www.cs.ualberta.ca/∼yaki`

use of this constraint allows muscle contractions in one direction to generate forces acting in perpendicular directions.

Due to their unique properties, understanding the principles governing the movement and control of the Octopus arm and other muscular hydrostats is of great interest to both physiologists and robotics engineers. Recent physiological and behavioral studies produced some interesting insights to the way the Octopus plans and controls its movements. Gutfreund et al. [3] investigated the reaching movement of an Octopus arm and showed that the motion is performed by a stereotypical forward propagation of a bend point along the arm. Yekutieli et al. [4] propose that the complex behavioral movements of the Octopus are composed from a limited number of "motion primitives", which are spatio-temporally combined to produce the arm's motion.

Although physical implementations of robotic arms based on the same principles are not yet available, recent progress in the technology of "artificial muscles" using electroactive polymers [5] may allow the construction of such arms in the near future. Needless to say, even a single such arm poses a formidable control challenge, which does not appear to be amenable to conventional control theoretic or robotics methodology. In this paper we propose a learning approach for tackling this problem. Specifically, we formulate the task of bringing some part of the arm into a goal region as a reinforcement learning (RL) problem. We then proceed to solve this problem using Gaussian process temporal difference learning (GPTD) algorithms [6, 7, 8].

## 2 The Domain

Our experimental test-bed is a finite-elements computer simulation of a planar variant of the Octopus arm, described in [9, 4]. This model is based on a decomposition of the arm into quadrilateral compartments, and the constant muscular volume constraint mentioned above is translated into a constant area constraint on each compartment. Muscles are modeled as dampened springs and the mass of each compartment is concentrated in point masses located at its corners[1]. Although this is a rather crude approximation of the real arm, even for a modest 10-segment model there are already 88 continuous state variables[2], making this a rather high dimensional learning problem. Figure 1 illustrates this model.

Since our model is 2–dimensional, all force vectors lie on the $x - y$ plane, and the arm's motion is planar. This limitation is due mainly to the high computational cost of the full 3–dimensional calculations for any arm of reasonable size. There are four types of forces acting on the arm: 1) The internal forces generated by the arm's muscles, 2) the vertical forces caused by the influence of gravity and the arm's buoyancy in the medium in which it is immersed (typically sea water), 3) drag forces produced by the arm's motion through this medium, and 4) internal pressure-induced forces responsible for maintaining the constant volume of each compartment. The use of simulation allows us to easily investigate different operating scenarios, such as zero or low gravity scenarios, different media, such as water, air or vacuum, and different muscle models. In this study, we used a simple linear model for the muscles. The force applied by a muscle at any given time $t$ is

$$F(t) = \big(k_0 + (k_{max} - k_0)A(t)\big)\big(\ell(t) - \ell_{rest}\big) + c\frac{d\ell(t)}{dt}.$$

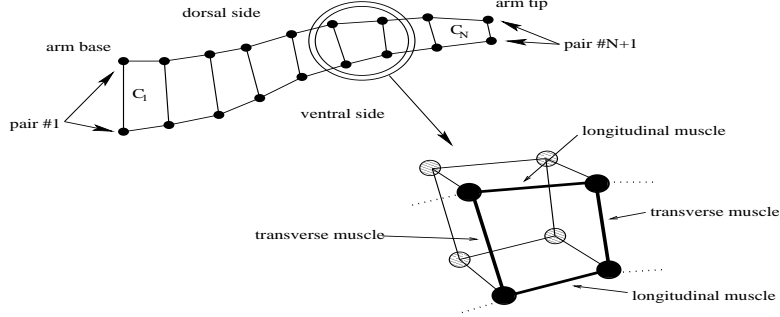

Figure 1: An $N$ compartment simulated Octopus arm. Each constant area compartment $C_i$ is defined by its surrounding 2 longitudinal muscles (ventral and dorsal) and 2 transverse muscles. Circles mark the $2N + 2$ point masses in which the arm's mass is distributed. In the bottom right one compartment is magnified with additional detail.

This equation describes a dampened spring with a controllable spring constant. The spring's length at time $t$ is $\ell(t)$, its resting length, at which it does not apply any force is $\ell_{rest}$.[3] The spring's stiffness is controlled by the activation variable $A(t) \in [0, 1]$. Thus, when the activation is zero, and the contraction is isometric (with zero velocity), the relaxed muscle exhibits a baseline passive stiffness $k_0$. In a fully activated isometric contraction the spring constant becomes $k_{max}$. The second term is a dampening, energy dissipating term, which is proportional to the rate of change in the spring's length, and (with $c > 0$) is directed to resist that change. This is a very simple muscle model, which has been chosen mainly due to its low computational cost, and the relative ease of computing the energy expended by the muscle (why this is useful will become apparent in the sequel). More complex muscle models can be easily incorporated into the simulator, but may result in higher computational overhead. For additional details on the modeling of the other forces and on the derivation of the equations of motion, refer to [4].

## 3   The Learning Algorithms

As mentioned above, we formulate the problem of controlling our Octopus arm as a RL problem. We are therefore required to define a Markov decision process (MDP), consisting of state and action spaces, a reward function and state transition dynamics. The states in our model are the Cartesian coordinates of the point masses and their first time-derivatives. A finite (and relatively small) number of actions are defined by specifying, for each action, a set of activations for the arm's muscles. The actions used in this study are depicted in Figure 2. Given the arm's current state and the chosen action, we use the simulator to compute the arm's state after a small fixed time interval. Throughout this interval the activations remain fixed, until a new action is chosen for the next interval. The reward is defined as $-1$ for non-goal states, and 10 for goal states. This encourages the controller to find policies that bring the arm to the goal as quickly as possible. In addition, in order to encourage smoothness and economy in the arm's movements, we subtract an energy penalty term from these rewards. This term is proportional to the total energy expended by all muscles during each action interval. Training is performed in an episodic manner: Upon reaching a goal, the current episode terminates and the arm is placed in a new initial position to begin a new episode. If a goal is not reached by some fixed amount of time, the

episode terminates regardless.

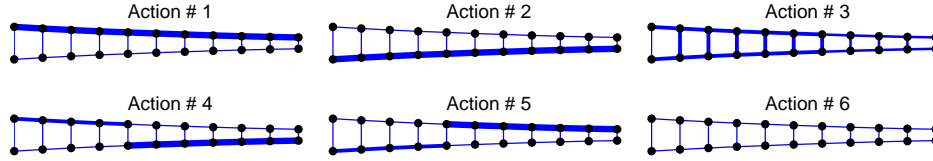

Figure 2: The actions used in the fixed-base experiments. Line thickness is proportional to activation intensity. For the rotating base experiment, these actions were augmented with versions of actions 1, 2, 4 and 5 that include clockwise and anti-clockwise torques applied to the arm's base.

The RL algorithms implemented in this study belong to the Policy Iteration family of algorithms [10]. Such algorithms require an algorithmic component for estimating the mean sum of (possibly discounted) future rewards collected along trajectories, as a function of the trajectory's initial state, also known as the *value function*. The best known RL algorithms for performing this task are *temporal difference* algorithms. Since the state space of our problem is very large, some form of function approximation must be used to represent the value estimator. Temporal difference methods, such as TD($\lambda$) and LSTD($\lambda$), are provably convergent when used with linearly parametrized function approximation architectures [10]. Used this way, they require the user to define a fixed set of basis functions, which are then linearly combined to approximate the value function. These basis functions must be defined over the entire state space, or at least over the subset of states that might be reached during learning. When local basis functions are used (e.g., RBFs or tile codes [11]), this inevitably means an exponential explosion of the number of basis functions with the dimensionality of the state space. Nonparametric GPTD learning algorithms[4] [8], offer an alternative to the conventional parametric approach. The idea is to define a nonparametric statistical generative model connecting the hidden values and the observed rewards, and a prior distribution over value functions. The GPTD modeling assumptions are that both the prior and the observation-noise distributions are Gaussian, and that the model equations relating values and rewards have a special linear form. During or following a learning session, in which a sequence of states and rewards are observed, Bayes' rule may be used to compute the posterior distribution over value functions, conditioned on the observed reward sequence. Due to the GPTD model assumptions, this distribution is also Gaussian, and is derivable in closed form. The benefits of using (nonparametric) GPTD methods are that 1) the resulting value estimates are generally not constrained to lie in the span of any predetermined set of basis functions, 2) no resources are wasted on unvisited state and action space regions, and 3) rather than the point estimates provided by other methods, GPTD methods provide complete probability distributions over value functions.

In [6, 7, 8] it was shown how the computation of the posterior value GP moments can be performed sequentially and online. This is done by a employing a forward selection mechanism, which is aimed at attaining a sparse approximation of the posterior moments, under a constraint on the resulting error. The input samples (states, or state-action pairs) used in this approximation are stored in a *dictionary*, the final size of which is often a good indicator of the problem's complexity. Since nonparametric GPTD algorithms belong to the family of kernel machines, they require the user to define a kernel function, which encodes her prior knowledge and beliefs concerning similarities and correlations in the domain at hand. More specifically, the kernel function $k(\cdot, \cdot)$ defines the *prior covariance* of the value process. Namely, for two arbitrary states $\mathbf{x}$ and $\mathbf{x}'$, $\mathbf{Cov}[V(\mathbf{x}), V(\mathbf{x}')] = k(\mathbf{x}, \mathbf{x}')$ (see [8] for details). In this study we experimented with several kernel functions, however, in this

paper we will describe results obtained using a third degree polynomial kernel, defined by $k(\mathbf{x}, \mathbf{x}') = (\mathbf{x}^\top \mathbf{x}' + 1)^3$. It is well known that this kernel induces a feature space of monomials of degree 3 or less [12]. For our 88 dimensional input space, this feature space is spanned by a basis consisting of $\binom{91}{3} = 121{,}485$ linearly independent monomials.

We experimented with two types of policy-iteration based algorithms. The first was optimistic policy iteration (OPI), in which, in any given time-step, the current GPTD value estimator is used to evaluate the successor states resulting from each one of the actions available at the current state. Since, given an action, the dynamics are deterministic, we used the simulation to determine the identity of successor states. An action is then chosen according to a semi-greedy selection rule (more on this below). A more disciplined approach is provided by a *paired actor-critic* algorithm. Here, two independent GPTD estimators are maintained. The first is used to determine the policy, again, by some semi-greedy action selection rule, while its parameters remain fixed. In the meantime, the second GPTD estimator is used to evaluate the stationary policy determined by the first. After the second GPTD estimator is deemed sufficiently accurate, as indicated by the GPTD value variance estimate, the roles are reversed. This is repeated as many times as required, until no significant improvement in policies is observed.

Although the latter algorithm, being an instance of approximate policy iteration, has a better theoretical grounding [10], in practice it was observed that the GPTD-based OPI worked significantly faster in this domain. In the experiments reported in the next section we therefore used the latter. For additional details and experiments refer to [13]. One final wrinkle concerns the selection of the initial state in a new episode. Since plausible arm configurations cannot be attained by randomly drawing 88 state variable from some simple distribution, a more involved mechanism for setting the initial state in each episode has to be defined. The method we chose is tightly connected to the GPTD mode of operation: At the end of each episode, 10 random states were drawn from the GPTD dictionary. From these, the state with the highest posterior value variance estimate was selected as the initial state of the next episode. This is a form of *active learning*, which is made possible by employing GPTD, and that is applicable to general episodic RL problems.

## 4 Experiments

The experiments described in this section are aimed at demonstrating the applicability of GPTD-based algorithms to large-scale RL problems, such as our Octopus arm. In these experiments we used the simulated 10-compartment arm described in Section 2. The set of goal states consisted of a circular region located somewhere within the potential reach of the arm (recall that the arm has no fixed length). The action set depends on the task, as described in Figure 2. Training episode duration was set to 4 seconds, and the time interval between action decisions was 0.4 seconds. This allowed a maximum of 10 learning steps per trial. The discount factor was set to 1.

The exploration policy used was the ubiquitous $\varepsilon$-greedy policy: The greedy action (i.e. the one for which the sum of the reward and the successor state's estimated value is the highest) is chosen with probability $1 - \varepsilon$, and with probability $\varepsilon$ a random action is drawn from a uniform distribution over all other actions. The value of $\varepsilon$ is reduced during learning, until the policy converges to the greedy one. In our implementation, in each episode, $\varepsilon$ was dependent on the number of successful episodes experienced up to that point. The general form of this relation is $\varepsilon = \varepsilon_0 N_{\frac{1}{2}} / (N_{\frac{1}{2}} + N_{goals})$, where $N_{goals}$ is the number of successful episodes, $\varepsilon_0$ is the initial value of $\varepsilon$ and $N_{\frac{1}{2}}$ is the number of successful episodes required to reduce $\varepsilon$ to $\varepsilon_0/2$.

In order to evaluate the quality of learned solutions, 100 initial arm configurations were cre-

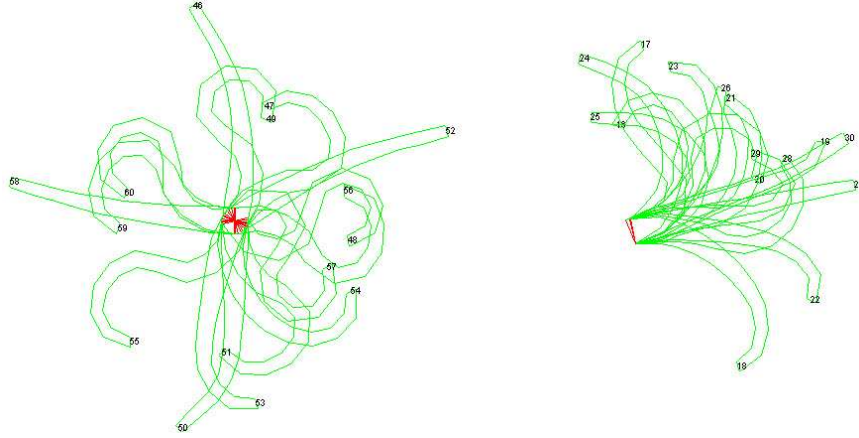

Figure 3: Examples of initial states for the rotating-base experiments (left) and the fixed-base experiments (right). Starting states also include velocities, which are not shown.

ated. This was done by starting a simulation from some fixed arm configuration, performing a long sequence of random actions, and sampling states randomly from the resulting trajectory. Some examples of such initial states are depicted in Figure 3. During learning, following each training episode, the GPTD-learned parameters were recorded on file. Each set of GPTD parameters defines a value estimator, and therefore also a greedy policy with respect to the posterior value mean. Each such policy was evaluated by using it, starting from each of the 100 initial test states. For each starting state, we recorded whether or not a goal state was reached within the episode's time limit (4 seconds), and the duration of the episode (successful episodes terminate when a goal state is reached). These two measures of performance were averaged over the 100 starting states and plotted against the episode index, resulting in two corresponding learning curves for each experiment[5].

We started with a simple task in which reaching the goal is quite easy. Any point of the arm entering the goal circle was considered as a success. The arm's base was fixed and the gravity constant was set to zero, corresponding to a scenario in which the arm moves on a horizontal frictionless plane. In the second experiment the task was made a little more difficult. The goal was moved further away from the base of the arm. Moreover, gravity was set to its natural level, of $9.8\frac{m}{s^2}$, with the motion of the arm now restricted to a vertical plane. The learning curves corresponding to these two experiments are shown in Figure 4. A success rate of 100% was reached after 10 and 20 episodes, respectively. In both cases, even after a success rate of 100% is attained, the mean time-to-goal keeps improving. The final dictionaries contained about 200 and 350 states, respectively.

In our next two experiments, the arm had to reach a goal located so that it cannot be reached unless the base of the arm is allowed to rotate. We added base-rotating actions to the basic actions used in the previous experiments (see Figure 2 for an explanation). Allowing a rotating base significantly increases the size of the action set, as well the size of the reachable state space, making the learning task considerably more difficult. To make things even more difficult, we rewarded the arm only if it reached the goal with its tip, i.e. the two point-masses at the end of the arm. In the first experiment in this series, gravity was switched on. A 99% success rate was attained after 270 trials, with a final dictionary size of

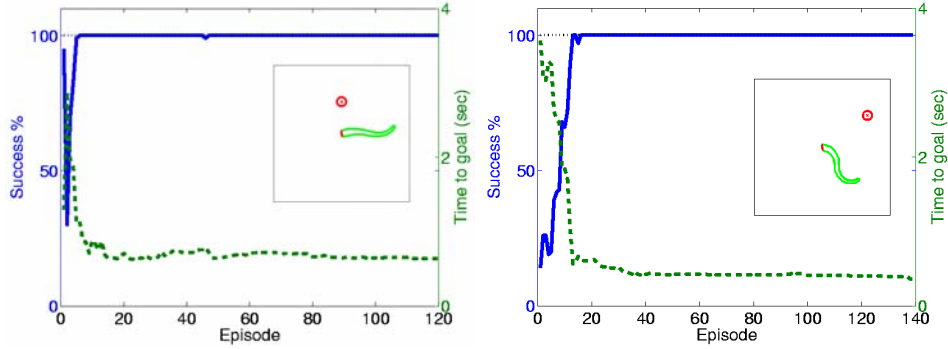

Figure 4: Success rate (solid) and mean time to goal (dashed) for a fixed-base arm in zero gravity (left), and with gravity (right). 100% success was reached after 10 and 20 trials, respectively. The insets illustrate one starting position and the location of the goal regions, in each case.

about 600 states. In the second experiment gravity was switched off, but a circular region of obstacle states was placed between the arm's base and the goal circle. If any part of the arm touched the obstacle, the episode immediately terminated with a negative reward of -2. Here, the success rate peaked at 40% after around 1000 episodes, and remained roughly constant thereafter. It should be taken into consideration that at least some of the 100 test starting states are so close to the obstacle that, regardless of the action taken, the arm cannot avoid hitting the obstacle. The learning curves are presented in Figure 5.

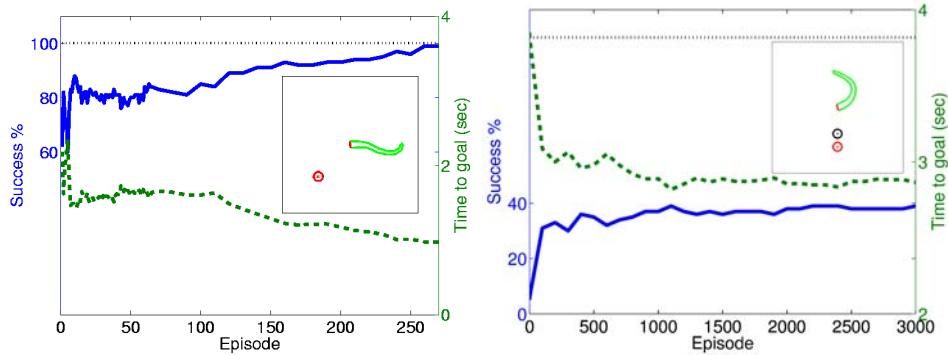

Figure 5: Success rate (solid) and mean time to goal (dashed) for a rotating-base arm with gravity switched on (left), and with gravity switched off but with an obstacle blocking the direct path to the goal (right). The arm has to rotate its base in order to reach the goal in either case (see insets). Positive reward was given only for arm-tip contact, any contact with the obstacle terminated the episode with a penalty. A 99% success rate was attained after 270 episodes for the first task, whereas for the second task success rate reached 40%.

Video movies showing the arm in various scenarios are available at `www.cs.ualberta.ca/~yaki/movies/`.

## 5   Discussion

Up to now, GPTD based RL algorithms have only been tested on low dimensional problem domains. Although kernel methods have handled high-dimensional data, such as handwrit-

ten digits, remarkably well in supervised learning domains, the applicability of the kernel-based GPTD approach to high dimensional RL problems has remained an open question. The results presented in this paper are, in our view, a clear indication that GPTD methods are indeed scalable, and should be considered seriously as a possible solution method by practitioners facing large-scale RL problems. Further work on the theory and practice of GPTD methods is called for. Standard techniques for model selection and tuning of hyper-parameters can be incorporated straightforwardly into GPTD algorithms. Value iteration-based variants, i.e. "GPQ-learning", would provide yet another useful set of tools.

The Octopus arm domain is of independent interest, both to physiologists and robotics engineers. The fact that reasonable controllers for such a complex arm can be learned from trial and error, in a relatively short time, should not be understated. Further work in this direction should be aimed at extending the Octopus arm simulation to a full 3-dimensional model, as well as applying our RL algorithms to real robotic arms based on the muscular hydrostat principle, when these become available.

### Acknowledgments

Y. E. was partially supported by the AICML and the Alberta Ingenuity fund. We would also like to thank the Ollendorff Minerva Center, for supporting this project.

## Footnotes

[1]For the purpose of computing volumes, masses, friction and muscle strength, the arm is effectively defined in three dimensions. However, no forces or motion are allowed in the third dimension. We also ignore the suckers located along the ventral side of the arm, and treat the arm as if it were symmetric with respect to reflection along its long axis. Finally, we comment that this model is restricted to modeling the mechanics of the arm and does not attempt to model its nervous system.

[2]10 segments result in 22 point masses, each being described by 4 state variables – the $x$ and $y$ coordinates and their respective first time-derivatives.

[3]It is assumed that at all times $\ell(t) \geq \ell_{rest}$. This is meant to ensure that our muscles can only apply force by contracting, as real muscles do. This can be assured by endowing the compartments with sufficiently high volumes, or equivalently, by setting $\ell_{rest}$ sufficiently low.

[4]GPTD models can also be defined parametrically, see [8].

[5]It is worth noting that this evaluation procedure requires by far more time than the actual learning, since each point in the graphs shown below requires us to perform 100 simulation runs. Whereas learning can be performed almost in real-time (depending on dictionary size), computing the statistics for a single learning run may take a day, or more.

## References

[1] G. Fiorito, C. V. Planta, and P. Scotto. Problem solving ability of Octopus Vulgaris Lamarck (Mollusca, Cephalopoda). *Behavioral and Neural Biology*, 53 (2):217–230, 1990.

[2] W.M. Kier and K.K. Smith. Tongues, tentacles and trunks: The biomechanics of movement in muscular-hydrostats. *Zoological Journal of the Linnean Society*, 83:307–324, 1985.

[3] Y. Gutfreund, T. Flash, Y. Yarom, G. Fiorito, I. Segev, and B. Hochner. Organization of Octopus arm movements: A model system for studying the control of flexible arms. *The journal of Neuroscience*, 16:7297–7307, 1996.

[4] Y. Yekutieli, R. Sagiv-Zohar, R. Aharonov, Y. Engel, B. Hochner, and T. Flash. A dynamic model of the Octopus arm. I. Biomechanics of the Octopus reaching movement. *Journal of Neurophysiology (in press)*, 2005.

[5] Y. Bar-Cohen, editor. *Electroactive Polymer (EAP) Actuators as Artificial Muscles - Reality, Potential and Challenges*. SPIE Press, 2nd edition, 2004.

[6] Y. Engel, S. Mannor, and R. Meir. Bayes meets Bellman: The Gaussian process approach to temporal difference learning. In *Proc. of the 20th International Conference on Machine Learning*, 2003.

[7] Y. Engel, S. Mannor, and R. Meir. Reinforcement learning with Gaussian processes. In *Proc. of the 22nd International Conference on Machine Learning*, 2005.

[8] Y. Engel. *Algorithms and Representations for Reinforcement Learning*. PhD thesis, The Hebrew University of Jerusalem, 2005. www.cs.ualberta.ca/~yaki/papers/thesis.ps.

[9] R. Aharonov, Y. Engel, B. Hochner, and T. Flash. A dynamical model of the octopus arm. In *Neuroscience letters. Supl. 48. Proceedings of the 6th annual meeting of the Israeli Neuroscience Society*, 1997.

[10] D.P. Bertsekas and J.N. Tsitsiklis. *Neuro-Dynamic Programming*. Athena Scientific, 1996.

[11] R.S. Sutton and Andrew G. Barto. *Reinforcement Learning: An Introduction*. MIT Press, 1998.

[12] J. Shawe-Taylor and N. Cristianini. *Kernel Methods for Pattern Analysis*. Cambridge University Press, Cambridge, England, 2004.

[13] Y. Engel, P. Szabo, and D. Volkinshtein. Learning to control an Octopus arm with Gaussian process temporal difference methods. Technical report, Technion Institute of Technology, 2005. www.cs.ualberta.ca/~yaki/reports/octopus.pdf.
